# Fast Resampling Weighted $v$-Statistics

**Chunxiao Zhou**
Mark O. Hatfield Clinical Research Center
National Institutes of Health
Bethesda, MD 20892
chunxiao.zhou@nih.gov

**Jiseong Park**
Dept of Math
George Mason Univ
Fairfax, VA 22030
jiseongp@gmail.com

**Yun Fu**
Dept of ECE
Northeastern Univ
Boston, MA 02115
yunfu@ece.neu.edu

## Abstract

In this paper, a novel and computationally fast algorithm for computing weighted $v$-statistics in resampling both univariate and multivariate data is proposed. To avoid any real resampling, we have linked this problem with finite group action and converted it into a problem of orbit enumeration. For further computational cost reduction, an efficient method is developed to list all orbits by their symmetry orders and calculate all index function orbit sums and data function orbit sums recursively. The computational complexity analysis shows reduction in the computational cost from $n!$ or $n^n$ level to low-order polynomial level.

## 1 Introduction

Resampling methods (e.g., bootstrap, cross-validation, and permutation) [3,5] are becoming increasingly popular in statistical analysis due to their high flexibility and accuracy. They have been successfully integrated into most research topics in machine learning, such as feature selection, dimension reduction, supervised learning, unsupervised learning, reinforcement learning, and active learning [2, 3, 4, 7, 9, 11, 12, 13, 20].

The key idea of resampling is to generate the empirical distribution of a test statistic by resampling with or without replacement from the original observations. Then further statistical inference can be conducted based on the empirical distribution, i.e., resampling distribution. One of the most important problems in resampling is calculating resampling statistics, i.e., the expected values of test statistics under the resampling distribution, because resampling statistics are compact representatives of the resampling distribution. In addition, a resampling distribution may be approximated by a parametric model with some resampling statistics, for example, the first several moments of a resampling distribution [5, 16]. In this paper, we focus on computing resampling weighted $v$-statistics [18] (see Section 2 for the formal definition). Suppose our data includes $n$ observations, a weighted $v$-statistic is a summation of products of data function terms and index function terms, i.e., weights, over all possible $k$ observations chosen from $n$ observations, where $k$ is the order of the weighted $v$-statistic. If we treat our data as points in a multi-dimensional space, a weighted $v$-statistic can be considered as an average of all possible weighted $k$-points distances. The higher $k$, the more complicated interactions among observations can be modeled in the weighted $v$-statistic. Machine learning researchers have already used weighted $v$-statistics in hypothesis testing, density estimation, dependence measurement, data pre-processing, and classification [6, 14, 19, 21] .

Traditionally, estimation of resampling statistics is solved by random sampling since exhaustive examination of the resampling space is usually ill advised [5,16]. There is a tradeoff between accuracy and computational cost with random sampling. To date, there is no systematic and efficient solution to the issue of exact calculation of resampling statistics. Recently, Zhou et.al. [21] proposed a recursive method to derive moments of permutation distributions (i.e., empirical distribution generated by resampling without replacement). The key strategy is to divide the whole index set (i.e., indices of all possible $k$ observations ) into several permutation equivalent index subsets such that the summa-

tion of the data/index function term over all permutations is invariant within each subset and can be calculated without conducting any permutation. Therefore, moments are obtained by summing up several subtotals. However, methods for listing all permutation equivalent index subsets and calculating of the respective cardinalities were not emphasized in the previous publication [21]. There is also no systematic way to obtain coefficients in the recursive relationship. Even only for calculating the first four moments of a second order resampling weighted $v$ statistic, hundreds of index subsets and thousands of coefficients have to be derived manually. The manual derivation is very tedious and error-prone. In addition, Zhou's work is limited to permutation (resampling without replacement) and is not applicable to bootstrapping (resampling with replacement) statistics.

In this paper, we propose a novel and computationally fast algorithm for computing weighted $v$-statistics in resampling both univariate and multivariate data. In the proposed algorithm, the calculation of weighted $v$-statistics is considered as a summation of products of data function terms and index function terms over a high-dimensional index set and all possible resamplings with or without replacement. To avoid any resampling, we link this problem with finite group actions and convert it into a problem of orbit enumeration [10]. For further computational cost reduction, an efficient method has been developed to list all orbits by their symmetry order and to calculate all index function orbit sums and data function orbit sums recursively. With computational complexity analysis, we have reduced the computational cost from $n!$ or $n^n$ level to low-order polynomial level. Detailed proofs have been included in the supplementary material.

In comparison with previous work [21], this study gives a theoretical justification of the permutation equivalence partition idea and extends it to other types of resamplings. We have built up a solid theoretical framework that explains the symmetry of resampling statistics using a product of several symmetric groups. In addition, by associating this problem with finite group action, we have developed an algorithm to enumerate all orbits by their symmetry order and generated a recursive relationship for orbits sum calculation systematically. This is a critical improvement which makes the whole method fully programmable and frees ourselves from onerous derivations in [21].

## 2 Basic idea

In general, people prefer choosing statistics which have some symmetric properties. All resampling strategies, such as permutation and bootstrap, are also more or less symmetric. These facts motivated us to reduce the computational cost by using abstract algebra.

This study is focused on computing resampling weighted $v$-statistics, i.e., $T(x) = \sum_{i_1=1}^{n} \cdots \sum_{i_d=1}^{n} w(i_1, \cdots, i_d) h(x_{i_1}, \cdots x_{i_d})$, where $x = (x_1, x_2, \cdots, x_n)^T$ is a collection of $n$ observations (univariate/multivariate), $w$ is an index function of $d$ indices, and $h$ is a data function of $d$ observations. Both $w$ and $h$ are symmetric, i.e., invariant under permutations of the order of variables. Weighted $v$-statistics cover a large amount of popular statistics. For example, in the case of multiple comparisons, observations are collected from $g$ groups: first group $(x_1, \cdots, x_{n_1})$, second group $(x_{n_1+1}, \cdots, x_{n_1+n_2})$, and last group $(x_{n-n_g+1}, \cdots, x_n)$, where $n_1, n_2, \cdots, n_g$ are numbers of observations in each group. In order to test the difference among groups, it is common to use the modified $F$ test statistic $T(x) = (\sum_{i=1}^{n_1} x_i)^2/n_1 + (\sum_{i=n_1+1}^{n_1+n_2} x_i)^2/n_2 + \cdots + (\sum_{i=n-n_g+1}^{n} x_i)^2/n_g$, where $n = n_1 + n_2 + \cdots + n_g$. We can rewrite the modified $F$ statistic [3] as a second order weighted $v$-statistic, i.e., $T(x) = \sum_{i_1=1}^{n} \sum_{i_2=1}^{n} w(i_1, i_2) h(x_{i_1}, x_{i_2})$, here $h(x_{i_1}, x_{i_2}) = x_{i_1} x_{i_2}$ and $w(i_1, i_2) = 1/n_k$ if both $x_{i_1}$ and $x_{i_2}$ belong to the $k$-th group, and $w(i_1, i_2) = 0$ otherwise.

The $r$-th moment of a resampling weighted $v$-statistic is:

$$
\begin{aligned}
E_\sigma\Big(T^r(x)\Big) &= E_\sigma\Big( \sum_{i_1,\cdots,i_d} w(i_1, \cdots, i_d) h(x_{\sigma \cdot i_1}, \cdots, x_{\sigma \cdot i_d}) \Big)^r \\
&= E_\sigma\bigg\{ \sum_{i_1^1,\cdots,i_d^1,\cdots,i_1^r,\cdots,i_d^r} \Big\{ \Big(\prod_{k=1}^{r} w(i_1^k, \cdots, i_d^k)\Big) \Big(\prod_{k=1}^{r} h(x_{\sigma \cdot i_1^k}, \cdots, x_{\sigma \cdot i_d^k})\Big) \Big\} \bigg\} \\
&= \frac{1}{|R|} \sum_{\sigma \in R} \bigg\{ \sum_{i_1^1,\cdots,i_d^1,\cdots,i_1^r,\cdots,i_d^r} \Big\{ \Big(\prod_{k=1}^{r} w(i_1^k, \cdots, i_d^k)\Big) \Big(\prod_{k=1}^{r} h(x_{\sigma \cdot i_1^k}, \cdots, x_{\sigma \cdot i_d^k})\Big) \Big\} \bigg\}, \quad (1)
\end{aligned}
$$

where $\sigma$ is a resampling which is uniformly distributed in the whole resampling space $R$. $|R|$, the number of all possible resamplings, is equal to $n!$ or $n^n$ for resampling without or with replacement. Thus the $r$-th moment of a resampling weighted $v$-statistic can be considered as a summation of products of data function terms and index function terms over a high-dimensional index set $U_d^r = \{1, \cdots, n\}^{dr}$ and all possible resamplings in $R$. Since both index space and resampling space are huge, it is computationally expensive for calculating resampling statistics directly.

For terminology convenience, $\{(i_1^1, \cdots, i_d^1), \cdots, (i_1^r, \cdots, i_d^r)\}$ is called an index paragraph, which includes $r$ index sentences $(i_1^k, \cdots, i_d^k), k = 1, \cdots, r$, and each index sentence has $d$ index words $i_j^k, j = 1, \cdots, d$. Note that there are three different types of symmetry in computing resampling weighted $v$-statistics. The first symmetry is that permutation of the order of index words will not affect the result since the data function is assumed to be symmetric. The second symmetry is the permutation of the order of index sentences since multiplication is commutative. The third symmetry is that each possible resampling is equally likely to be chosen.

In order to reduce the computational cost, first, the summation order is exchanged,

$$E_\sigma\left(T^r(x)\right) = \sum_{i_1^1, \cdots, i_d^1, \cdots, i_1^r, \cdots, i_d^r} \left\{\left(\prod_{k=1}^{r} w(i_1^k, \cdots, i_d^k)\right) E_\sigma\left(\prod_{k=1}^{r} h(x_{\sigma \cdot i_1^k}, \cdots, x_{\sigma \cdot i_d^k})\right)\right\}, \quad (2)$$

where $E_\sigma\left(\prod_{k=1}^{r} h(x_{\sigma \cdot i_1^k}, \cdots, x_{\sigma \cdot i_d^k})\right) = \frac{1}{|R|} \sum_{\sigma \in R}\left(\prod_{k=1}^{r} h(x_{\sigma \cdot i_1^k}, \cdots, x_{\sigma \cdot i_d^k})\right)$.

The whole index set $U_d^r = \{1, \cdots, n\}^{dr} = \Big\{\{(i_1^1, \cdots, i_d^1), \cdots, (i_1^r, \cdots, i_d^r)\} | i_m^k \in \{1, \cdots, n\}; m = 1, \cdots, d; k = 1, \cdots, r\Big\}$ is then divided into disjoint index subsets, in which $E_\sigma\left(\prod_{k=1}^{r} h(x_{\sigma \cdot i_1^k}, \cdots, x_{\sigma \cdot i_d^k})\right)$ is invariant. The above index set partition simplifies the computing of resampling statistics in the following sense: (a) we only need to calculate $E_\sigma\left(\prod_{k=1}^{r} h(x_{\sigma \cdot i_1^k}, \cdots, x_{\sigma \cdot i_d^k})\right)$ once per each index subset, (b) due to the symmetry of resampling, the calculation of $E_\sigma\left(\prod_{k=1}^{r} h(x_{\sigma \cdot i_1^k}, \cdots, x_{\sigma \cdot i_d^k})\right)$ is equivalent to calculating the average of all data function terms within the corresponding index subset, then we can completely replace all resamplings with simple summations, and (c) for further computational cost reduction, we can sort all index subsets in their symmetry order and calculate all index subset summations recursively. We will discuss the details in the following sections for both resampling without or with replacement. The abstract algebra terms used in this paper are listed as follows.

**Terminology.** A group is a non-empty set $G$ with a binary operation satisfying the following axioms: closure, associativity, identity, and invertibility. The symmetric group on a set, denoted as $S_n$, is the group consisting of all bijections or permutations of the set. A semigroup has an associative binary operation defined and is closed with respect to this operation, but not all its elements need to be invertible. A monoid is a semigroup with an identity element. A set of generators is a subset of group elements such that all the elements in the group can be generated by repeated composition of the generators. Let $X$ be a set and $G$ be a group. A group action is a mapping $G \times X \to X$ which satisfies the following two axioms: (a) $e \cdot x \mapsto x$ for all $x \in X$, and (b) for all $a, b \in G$ and $x \in X$, $a \cdot (b \cdot x) = (ab) \cdot x$. Here the $'\cdot'$ denotes the action. It is well known that a group action defines an equivalence relationship on the set $X$, and thus provides a disjoint set partition on it. Each part of the set partition is called an orbit that denotes the trajectory moved by all elements within the group. We use symbol $[\,]$ to represent an orbit. Two elements, $x$ and $y \in X$ fall into the same orbit if there exists a $g \in G$ such that $x = g \cdot y$. The set of orbits is denoted by $G \backslash\!\backslash X$. A transversal of orbits is a set of representatives containing exactly one element from each orbit. In this paper, we limit our discussion to only finite groups [10,17].

## 3 Permutation

For permutation statistics, observations are permuted in all possible ways, i.e., $R = S_n$. Based on the three types of symmetry, we link the permutation statistics calculation with a group action.

**Definition 1.** The action of $G := S_n \times S_r \times S_d^r$ on the index set $U_d^r$ is defined as

$(\sigma, \tau, \pi_1, \cdots, \pi_r) \cdot i_m^k := \sigma \cdot i_{\pi_k^{-1} \cdot m}^{\tau^{-1} \cdot k}$, where $m \in \{1, \cdots, d\}$, and $k \in \{1, \cdots, r\}$.

Here, $\pi_k$ denotes the permutation of the order of index words within the $k$-th index sentence, $\tau$ denotes the permutation of the order of $r$ index sentences, and $\sigma$ denotes the permutation of the value of an index word from $1$ to $n$. For example, let $n = 4$, $d = 2$, $r = 2$, $\pi_1 = \pi_1^{-1} = 1 \rightarrow 2, 2 \rightarrow 1$, $\pi_2 = \pi_2^{-1} = 1 \rightarrow 1, 2 \rightarrow 2$, $\tau = \tau^{-1} = 1 \rightarrow 2, 2 \rightarrow 1$, and $\sigma = 1 \rightarrow 2, 2 \rightarrow 4, 3 \rightarrow 3, 4 \rightarrow 1$, then $(\sigma, \tau, \pi_1, \pi_2) \cdot \{(1,4)(3,4)\} = \{(3,1)(1,2)\}$ by $\{(1,4)(3,4)\} \rightarrow \{(4,1)(3,4)\} \rightarrow \{(3,4)(4,1)\} \rightarrow \{(3,1)(1,2)\}$. Note that the reason to define the action in this way is to guarantee $G \times U_d^r \rightarrow U_d^r$ is a group action.

In most applications, both $r$ and $d$ are much less than the sample size $n$, we assume throughout this paper that $n \gg dr$.

**Proposition 1.** The data function sum $E_\sigma\left(\prod_{k=1}^r h(x_{\sigma \cdot i_1^k}, \cdots, x_{\sigma \cdot i_d^k})\right)$ is invariant within each index orbit of group action $G := S_n \times S_r \times S_d^r$ acting on the index set $U_d^r$ as defined in definition 1, and $E_\sigma\left(\prod_{k=1}^r h(x_{\sigma \cdot i_1^k}, \cdots, x_{\sigma \cdot i_d^k})\right) =$

$$\sum_{\{(j_1^1, \cdots, j_d^1), \cdots, (j_1^r, \cdots, j_d^r)\} \in [\{(i_1^1, \cdots, i_d^1), \cdots, (i_1^r, \cdots, i_d^r)\}]} \frac{\prod_{k=1}^r h(x_{j_1^k}, \cdots, x_{j_d^k})}{\mathrm{card}\left([\{(i_1^1, \cdots, i_d^1), \cdots, (i_1^r, \cdots, i_d^r)\}]\right)}, \quad (3)$$

where $\mathrm{card}\left([\{(i_1^1, \cdots, i_d^1), \cdots, (i_1^r, \cdots, i_d^r)\}]\right)$ is the cardinality of the index orbit, i.e., the number of indices within the index orbit $[\{(i_1^1, \cdots, i_d^1), \cdots, (i_1^r, \cdots, i_d^r)\}]$.

Due to the invariance property of $E_\sigma\left(\prod_{k=1}^r h(x_{\sigma \cdot i_1^k}, \cdots, x_{\sigma \cdot i_d^k})\right)$, the calculation of permutation statistics can be simplified by summing up all index function product terms in each index orbit.

**Proposition 2.** The $r$-th moment of permutation statistics can be obtained by summing up the product of the data function orbit sum $h_\lambda$ and the index function orbit sum $w_\lambda$ over all index orbits,

$$E_\sigma\left(T^r(x)\right) = \sum_{\lambda \in L} \frac{w_\lambda h_\lambda}{\mathrm{card}([\lambda])}, \quad (4)$$

where $\lambda = \{(i_1^1, \cdots, i_d^1), \cdots, (i_1^r, \cdots, i_d^r)\}$ is a representative index paragraph, $[\lambda]$ is the index orbit including $\lambda$, and $L$ is a transversal of all index orbits . The data function orbit sum is

$$h_\lambda = \sum_{\{(j_1^1, \cdots, j_d^1), \cdots, (j_1^r, \cdots, j_d^r)\} \in [\lambda]} \prod_{k=1}^r h(x_{j_1^k}, \cdots, x_{j_d^k}), \quad (5)$$

and the index function orbit sum is

$$w_\lambda = \sum_{\{(j_1^1, \cdots, j_d^1), \cdots, (j_1^r, \cdots, j_d^r)\} \in [\lambda]} \prod_{k=1}^r w(j_1^k, \cdots, j_d^k). \quad (6)$$

Proposition 2 shows that the calculation of resampling weighted $v$-statistics can be solved by computing data function orbit sums, index function orbit sums, and cardinalities of all orbits defined in definition 1. We don't need to conduct any real permutation at all.

Now we demonstrate how to calculate orbit cardinalities, $h_\lambda$ and $w_\lambda$. The following shows a naive algorithm to enumerate all index paragraphs and cardinality of each orbit of $G \backslash\!\backslash U_d^r$, which are needed to calculate $h_\lambda$ and $w_\lambda$. We construct a Cayley Action Graph with a vertex set of all possible index paragraphs in $U_d^r$. We connect a directed edge from $\{(i_1^1, \cdots, i_d^1), \cdots, (i_1^r, \cdots, i_d^r)\}$ to $\{(j_1^1, \cdots, j_d^1), \cdots, (j_1^r, \cdots, j_d^r)\}$ if $\{(j_1^1, \cdots, j_d^1), \cdots, (j_1^r, \cdots, j_d^r)\} = g_k\{(i_1^1, \cdots, i_d^1), \cdots, (i_1^r, \cdots, i_d^r)\}$, where $g_k$ is a generator $\in \{g_1, \cdots, g_p\}$. $\{g_1, \cdots, g_p\}$ is the set of generators of group $G$, i.e., $G = \langle g_1, \cdots, g_p \rangle$. It is sufficient and efficient to use the set of generators of group to construct the Cayley Action Graph, instead of using the set of all group elements. For example, we can choose $\{g_1, \cdots, g_p\} = \{\sigma_1, \sigma_2\} \times \{\tau_1, \tau_2\} \times \{\pi_1, \pi_2\}^r$, where $\sigma_1 = (12 \cdots n)$, $\sigma_2 = (12)$, $\tau_1 = (12 \cdots r)$, $\tau_2 = (12)$, $\pi_1 = (12 \cdots d)$, and $\pi_2 = (12)$. Here $\sigma_1 = (12 \cdots n)$ denotes the permutation $1 \rightarrow 2, 2 \rightarrow 3, \cdots, n \rightarrow 1$, and $\sigma_2 = (12)$ denotes

$1 \to 2, 2 \to 1, 3 \to 3, \cdots, n \to n$. Note that listing the index paragraphs of each orbit is equivalent to finding all connected components in the Cayley Action Graph, which can be performed by using existing depth-first or breadth-first search methods [15]. Figure 1 demonstrates the Cayley Action Graph of $G \backslash\!\backslash U_2^1$, where $d = 2$, $r = 1$, and $n = 3$. Since the main effort here is to construct the Cayley Action Graph, the computational cost of the naive algorithm is $O(n^{dr}p) = O(n^{dr}2^{2+r})$. Moreover, the memory cost is $O(n^{dr})$. Unfortunately, this algorithm is not an offline one since we usually do not know the data size $n$ before we have the data at hand, even $d$ and $r$ can be preset. In other words, we can not list all index orbits before we know the data size $n$. Moreover, since $n^{dr}2^{2+r}$ is still computationally expensive, the naive algorithm is ill advised even if $n$ is preset.

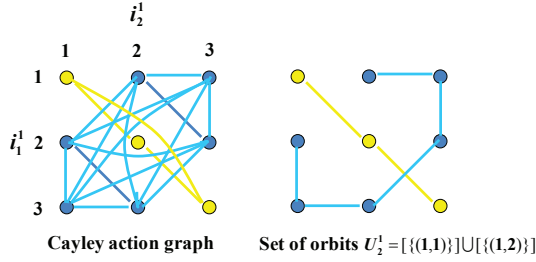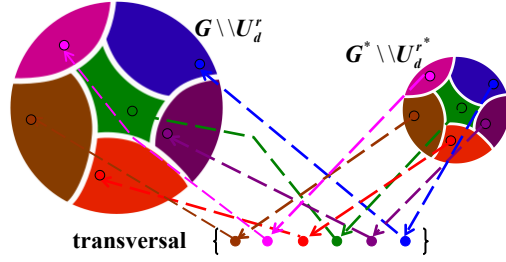

Figure 1: Cayley action graph for $G \backslash\!\backslash U_2^{1\,*}$.      Figure 2: Finding the transversal.

In table 1, we propose an improved offline algorithm in which we assume that $d$ and $r$ are preset. For computing $h_\lambda$ and $w_\lambda$, we find that we do not need to know all the index paragraphs within each index orbit. Since each orbit is well structured, it is enough to only list a transversal of orbits $G \backslash\!\backslash U_d^r$ and corresponding cardinalities. For example, there are two orbits, $[\{(1,1)\}]$ and $[\{(1,2)\}]$, when $d = 2$ and $r = 1$. $[\{(1,1)\}]$, with cardinality $n$, includes all index paragraphs with $i_1^1 = i_2^1$. $[\{(1,2)\}]$, with cardinality $n(n-1)$, includes all index paragraphs with $i_1^1 \neq i_2^1$. Actually, the transversal $L = \left\{\{(1,1)\}, \{(1,2)\}\right\}$ carries all the above information. This finding reduces the computation cost dramatically.

**Definition 2.** We define an index set $U_d^{r\,*} = \{1, \cdots, dr\}^{dr} = \Big\{\{(i_1^1, \cdots, i_d^1), \cdots, (i_1^r, \cdots, i_d^r)\} | i_m^k$

$\in \{1, \cdots, dr\}; m = 1, \cdots, d; k = 1, \cdots, r\Big\}$ and a group $G^* := S_{dr} \times S_r \times S_d{}^r$.

Since we assumed $n \gg dr$, $U_d^{r\,*}$ is a subset of the index set $U_d^r$. The group $G^*$ can be considered a subgroup of $G$ since the group $S_{dr}$ can be naturally embedded into the group $S_n$. Both $U_d^{r\,*}$ and $G^*$ are unrelated to the sample size $n$.

**Proposition 3.** The transversal of $G^* \backslash\!\backslash U_d^{r\,*}$ is also a transversal of $G \backslash\!\backslash U_d^r$.

By proposition 3, we notice that the listing of the transversal of $G \backslash\!\backslash U_d^r$ is equivalent to the listing of the transversal of $G^* \backslash\!\backslash U_d^{r\,*}$(see Figure 2). The latter is computationally much easier than the former since the cardinalities of $G^*$ and $U_d^{r\,*}$ are much smaller than those of $G$ and $U_d^r$ when $n \gg dr$. Furthermore, finding the transversal of $G^* \backslash\!\backslash U_d^{r\,*}$ can be done without knowning sample size $n$. Due to the structure of each orbit of $G \backslash\!\backslash U_d^r$, we can calculate the cardinality of each orbit of $G \backslash\!\backslash U_d^r$ with the transversal of $G^* \backslash\!\backslash U_d^{r\,*}$, although $G \backslash\!\backslash U_d^r$ and $G^* \backslash\!\backslash U_d^{r\,*}$ have different caridnalities for corresponding orbits.

Table 1: Offline double sided searching algorithm for listing the transversal

| |
|---|
| Input: $d$ and $r$, |
| 1. Starting from an orbit representative $\{(1, \cdots, d), \cdots, ((r-1)d+1, \cdots, rd)\}$ |
| 2. Construct the transversal of $S_{dr} \backslash\!\backslash U_d^{r\,*}$ by merging |
| 3. Construct the transversal of of $G^* \backslash\!\backslash U_d^{r\,*}$ by graph isomorphism testing |
| 4. Ending to an orbit representative $\{(1, \cdots, 1), \cdots, (1, \cdots, 1)\}$ |
| Output: a transversal $L$ of $G \backslash\!\backslash U_d^r$, $\#(\lambda)$, $\#(\lambda \to \nu)$, and merging order(symmetry order) of orbits |

Comparing with the Cayley Action Graph naive algorithm, our improved algorithm lists the transversal of $G \backslash\!\backslash U_d^r$ and calculates the cardinalities of all orbits more efficiently. In addition, the improved algorithm also assigns a symmetry order to all orbits, which helps further reduce the computational

cost of the data function orbit sum $h_\lambda$ and the index function orbit sum $w_\lambda$. The base of our improved algorithm is on the fact that a subgroup acting on the same set causes a finer partition. On one hand, it is challenging to directly list the transversal of $G^* \backslash\backslash U_d^{r*}$. On the other hand, it is much easier to find two related group actions, causing finer and coarser partitions of $U_d^{r*}$. These two group actions help us find the transversal of $G^* \backslash\backslash U_d^{r*}$ efficiently with a double sided searching method.

**Definition 3.** The action of $S_{dr}$ on the index set $U_d^{r*}$ is defined as $\sigma \cdot i_m^k$, where $\sigma \in S_{dr}$, $m \in \{1, \cdots, d\}$, and $k \in \{1, \cdots, r\}$. Each orbit of $S_{dr} \backslash\backslash U_d^{r*}$ is denoted by $[\{(i_1^1, \cdots, i_d^1), \cdots, (i_1^r, \cdots, i_d^r)\}]^s$.

Note the group action defined in definition 3 only allows permutation of index values, it does not allow shuffling of index words within each index sentence or of index sentences. Since $S_{dr}$ is embedded in $G^*$, the set of orbits $S_{dr} \backslash\backslash U_d^{r*}$ is a finer partition of $G^* \backslash\backslash U_d^{r*}$. For example, both $[\{(1,2)(1,2)\}]^s$ and $[\{(1,2)(2,1)\}]^s$ are finer partitions of $[\{(1,2)(1,2)\}]$. In addition, it is easy to construct a transversal of $S_{dr} \backslash\backslash U_d^{r*}$ by merging distinct index elements.

**Definition 4.** Given a representative $I$, which includes at least two distinct index values, for example $i \neq j$, an operation called merging replaces all index values of $i$ or $j$ with $\min(i, j)$.

For example, $[\{(1,2)(2,3)\}]$ becomes $[\{(1,1)(1,3)\}]$ after merging the index values of 1 and 2.

**Definition 5.** The action of $S_{dr} \times S_{dr}$ on the index set $U_d^{r*}$ is defined as $\sigma \cdot i_{\theta^{-1} \cdot (k,m)_w}^{\theta^{-1} \cdot (k,m)_s}$, where $\theta \in S_{dr}$ denotes a permutation of all $dr$ index words without any restriction, i.e. $\theta^{-1} \cdot (k, m)_s$ denotes the index sentence location after permutation $\theta$, and $\theta^{-1} \cdot (k, m)_w$ denotes the index word location after permutation $\theta$. The orbit of $S_{dr} \times S_{dr} \backslash\backslash U_d^{r*}$ is denoted by $[\{(i_1^1, \cdots, i_d^1), \cdots, (i_1^r, \cdots, i_d^r)\}]^l$.

Since the group action defined in definition 5 allows free shuffling of the order of all $dr$ index words, the order does not matter for $S_{dr} \times S_{dr} \backslash\backslash U_d^{r*}$ and shuffling can across different sentences. For example, $[\{(1,2)(1,2)\}]^l = [\{(1,1)(2,2)\}]^l$. $S_{dr} \times S_{dr} \backslash\backslash U_d^{r*}$ is a coarser partition of $G^* \backslash\backslash U_d^{r*}$.

**Proposition 4.** A transversal of $S_{dr} \backslash\backslash U_d^{r*}$ can be generated by all possible mergings of $[\{(1, \cdots, d), \cdots, (d(r-1) + 1, \cdots, dr)\}]^s$.

**Proposition 5.** Enumerating a transversal of $S_{dr} \times S_{dr} \backslash\backslash U_d^{r*}$ is equivalent to the integer partition of $dr$.

We start the transversal graph construction from an initial orbit $[\{(1, \cdots, d), \cdots, (d(r-1) + 1, \cdots, dr)\}]^s$, i.e, all index elements have distinct values. Then we generate new orbits of $S_{dr} \backslash\backslash U_d^{r*}$ by merging distinct index values in existing orbits until we meet $[\{(1, \cdots, 1), \cdots, (1, \cdots, 1)\}]^s$, i.e., all index elements have equal values. We also add an edge from an existing orbit to a new orbit generated by merging the existing one. The procedure for $d = 2$, $r = 2$ case is shown in Figure 3.

Now we generate the transversal of $G^* \backslash\backslash U_d^{r*}$ from that of $S_{dr} \backslash\backslash U_d^{r*}$. This can be done by checking whether two orbits in $S_{dr} \backslash\backslash U_d^{r*}$ are equivalent in $G^* \backslash\backslash U_d^{r*}$. Actually, orbit equivalence checking is equivalent to the classical graph isomorphism problem since we can consider each index word as a vertex and connect two index words if they belong to the same index sentence.

The graph isomorphism testing can be done by Luks's famous algorithm [1,15] with computational cost $\exp\left(O(\sqrt{v \log v})\right)$, where $v$ is the number of vertices. Figure 4 shows a transversal of $G^* \backslash\backslash U_2^{2*}$ generated from that of $S_4 \backslash\backslash U_2^{2*}$ (Figure 3). By proposition 3, it is also a transversal of $G \backslash\backslash U_2^2$. Since $G^* \backslash\backslash U_d^{r*}$ is a finer partition of $S_{dr} \times S_{dr} \backslash\backslash U_d^{r*}$, orbit equivalence testing is only necessary when two orbits of $S_{dr} \backslash\backslash U_d^{r*}$ correspond to the same integer partition. This is why we named this algorithm double sided searching.

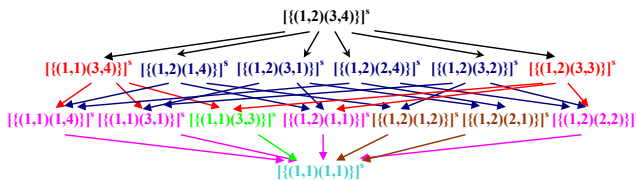

Figure 3: Transversal graph for $S_4 \backslash\backslash U_2^{2*}$.

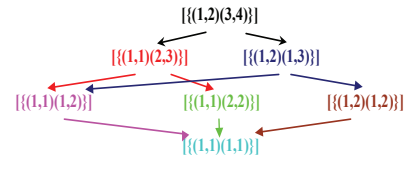

Figure 4: Transversal graph for $G \backslash\backslash U_2^2$.

**Definition 6.** For any two index orbit representatives $\lambda \in L$ and $\nu \in L$, we say that $\nu$ has a lower merging or symmetry order than that of $\lambda$, i.e., $\nu \prec \lambda$, if $[\nu]$ can be obtained from $[\lambda]$ by several mergings. Or there is a path from $[\lambda]$ to $[\nu]$ in the transversal graph. Here $L$ denotes a transversal set of all orbits.

**Definition 7.** We define $\#(\lambda)$ as the number of $S_{dr} \backslash\backslash U_d^{r\,*}$ orbits in $[\lambda]$. We also define $\#(\lambda \to \nu)$ as the number of different $[\nu]^s$s which can be reached from a $[\lambda]^s$.

It is easy to get $\#(\lambda)$ when we generate a transversal graph of $G \backslash\backslash U_d^r$ from that of $S_{dr} \backslash\backslash U_d^{r\,*}$. The $\#(\lambda \to \nu)$ can also be obtained from the transversal graph of $G \backslash\backslash U_d^r$ by counting the number of different $[\nu]^s$s which can be reached from a $[\lambda]^s$. For example, there are edges connecting $[\{(1,1)(3,4)\}]^s$ to $[\{(1,1)(1,4)\}]^s$ and $[\{(1,1)(3,1)\}]^s$. Since $[\{(1,1)(1,4)\}] = [\{(1,1)(3,1)\}] = [\{(1,1)(1,2)\}]$, $\#(\lambda = \{(1,1)(2,3)\} \to \nu = \{(1,1)(1,2)\}) = 2$. Note that this number can also be obtained from $[\{(1,2)(3,3)\}]^s$ to $[\{(1,2)(1,1)\}]^s$ and $[\{(1,2)(2,2)\}]^s$.

The difficulty for computing data function orbit sum and index function orbit sum comes from two constraints: equal constraint and unequal constraint. For example, in the orbit $[\{(1,1),(2,2)\}]$, the equal constraint is that the first and the second index values are equal and the third and fourth index values are also equal. On the other hand, the unequal constraint requires that the first two index values are different from the last two. Due to the difficulties mentioned, we solve this problem by first relaxing the unequal constraint and then applying the principle of inclusion and exclusion. Thus, the calculation of an orbit sum can be separated into two parts: the relaxed orbit sum without unequal constraint and lower order orbit sums. For example, the relaxed index function orbit sum is

$$w^*_{\lambda=[\{(1,1),(2,2)\}]} = \sum_{i,j} w(i,i)w(j,j) = \left( \sum_i w(i,i) \right)^2.$$

**Proposition 6.** The index function orbit sum $w_\lambda$ can be calculated by subtracting all lower order orbit sums from the corresponding relaxed index function orbit sum $w^*_\lambda$, i.e., $w_\lambda = w^*_\lambda - \sum_{\nu \prec \lambda} w_\nu \frac{\#(\lambda)}{\#(\nu)} \#(\lambda \to \nu)$. The cardinality of $[\lambda]$ is $\#(\lambda)n(n-1)\cdots(n-q+1)$, where $q$ is the number of distinct values in $\lambda$. The calculation of the data index function orbit sum $h_\lambda$ is similar.

So the computational cost mainly depends on the calculation of relaxed orbit sum and the lowest order orbit sum. The computational cost of the lowest order term is $O(n)$. The calculation of relaxed orbit can be done by Zhou's greedy graph search algorithm [21].

**Proposition 7.** For $d \geq 2$, let $m(m-1)/2 \leq rd(d-1)/2 < (m+1)m/2$, where $r$ is the order of moment and $m$ is an integer. For a $d$-th order weighted $v$-statistic, the computational cost of the orbit sum for the $r$-th moment is bounded by $O(n^m)$. When $d = 1$, the computational complexity of the orbit sum is $O(n)$.

## 4  Bootstrap

Since Bootstrap is resampling with replacement, we need to change $S_n$ to the set of all possible endofunctions $\text{End}_n$ in our computing scheme. In mathematics, an endofunction is a mapping of a set to its subset. With this change, $H := \text{End}_n \times S_r \times S_d^r$ acting on $U_d^r$ becomes a monoid action instead of a group action since endofunction is not invertible. The monoid action also divides the $U_d^r$ into several subsets. However, these subsets are not necessarily disjoint after mapping. For example, when $d = 2$ and $r = 1$, we can still divide the index set $U_2^1$ into two subsets, i.e., $[(1,1)]$ and $[(1,2)]$. However, $[(1,2)]$ is mapped to $U_2^1 = [(1,2)] \bigcup [(1,1)]$ by monoid action $H \times U_d^r \to U_d^r$, although $[(1,1)]$ is still mapped to itself. Fortunately, the computation of Bootstrap weighted $v$-statistics only needs index function orbit sums and relaxed data function orbit sums in the corresponding permutation computation. Therefore, the Bootstrap weighted $v$-statistics calculation is just a sub-problem of permutation weighted $v$-statistics calculation.

**Proposition 8.** We can obtain the $r$-th moment of bootstrapping weighted $v$-statistics by summing up the product of the index function orbit sum $w_\lambda$ and the relaxed data function orbit sum $h^*_\lambda$ over all index orbits, i.e.,

$$E_\sigma(T^r(x)) = \sum_{\lambda \in L} \frac{w_\lambda h^*_\lambda}{\text{card}([\lambda^*])}, \tag{7}$$

where $\sigma \in \text{End}_n$, $\text{card}([\lambda^*]) = \#(\lambda)n^q$, and $q$ is the number of distinct values in $\lambda$.

Table 2: Comparison of accuracy and complexity for calculation of resampling statistics.

| | | Methods | 2nd moment | 3rd | 4th | Time |
|---|---|---|---|---|---|---|
| Permutation | Linear | Exact | 0.7172 | -0.8273 | 1.0495 | 1.1153e3 |
| | | Our | 0.7172 | -0.8273 | 1.0495 | 0.0057 |
| | | Random | 0.7014 | -0.8326 | 1.0555 | 0.5605 |
| | Quadratic | Exact | 1.0611e3 | -4.6020e4 | 2.1560e6 | 1.718e3 |
| | | Our | 1.0611e3 | -4.6020e4 | 2.1560e6 | 0.006 |
| | | Random | 1.0569e3 | -4.5783e4 | 2.1825e6 | 2.405 |
| Bootstrap | Linear | Exact | 3.5166 | 8.9737 | 35.4241 | 204.4381 |
| | | Our | 3.5166 | 8.9737 | 35.4241 | 0.0053 |
| | | Random | 3.4769 | 8.8390 | 34.6393 | 0.3294 |
| | Quadratic | Exact | 2.4739e5 | -6.0322e6 | 2.6998e8 | 445.536 |
| | | Our | 2.4739e5 | -6.0322e6 | 2.6998e8 | 0.005 |
| | | Random | 2.4576e5 | -5.9825e6 | 2.6589e8 | 1.987 |

The computational cost of bootstrapping weighted $v$-statistics is the same level as that of permutation statistics.

# 5 Numerical results

To evaluate the accuracy and efficiency of our mothds, we generate simulated data and conduct permutation and bootstrapping for both linear test statistic $\sum_{i=1}^{n} w(i)h(x_i)$ and quadratic test statistic $\sum_{i_1=1}^{n} \sum_{i_2=1}^{n} w(i_1, i_2)h(x_{i_1}, x_{i_2})$ . To demonstrate the universal applicability of our method and prevent a chance result, we generate $w(i), h(x_i), w(i_1, i_2), h(x_{i_1}, x_{i_2})$ randomly. We compare the accuracy and complexity among exact permutation/bootstrap, random permutaton/bootrap (10,000 times), and our methods. Table 2 shows comparisons for computing the second, third, and fourth moments of permutation statistics with 11 observations (the running time is in seconds) and of bootstrap statistics with 8 observations.

In all cases, our method achieves the same moments as those of exact permutation/bootstrap, and reduces computational cost dramatically comparing with both random sampling and exact sampling. For demonstration purpose, we choose a small sample size here, i.e., sample size is 11 for permutation and 8 for bootstrap. Our method is expected to gain more computational efficiency as $n$ increases.

# 6 Conclusion

In this paper, we propose a novel and computationally fast algorithm for computing weighted $v$-statistics in resampling both univariate and multivariate data. Our theoretical framework reveals that the three types of symmetry in resampling weighted $v$-statistics can be represented by a product of symmetric groups. As an exciting result, we demonstrate the calculation of resampling weighted $v$-statistics can be converted into the problem of orbit enumeration. A novel efficient orbit enumeration algorithm has been developed by using a small group acting on a small index set. For further computational cost reduction, we sort all orbits by their symmetry order and calculate all index function orbit sums and data function orbit sums recursively. With computational complexity analysis, we have reduced the computational cost from $n!$ or $n^n$ level to low-order polynomial level.

# 7 Acknowledgement

This research was supported by the Intramural Research Program of the NIH, Clinical Research Center and through an Inter-Agency Agreement with the Social Security Administration, the NSF CNS 1135660, Office of Naval Research award N00014-12-1-0125, Air Force Office of Scienticfic Research award FA9550-12-1-0201, and IC Postdoctoral Research Fellowship award 2011-11071400006.

# References

[01] Babai, L., Kantor, W.M. , and Luks, E.M. (1983), Computational complexity and the classification of finite simple groups, Proc. 24th FOCS, pp. 162-171.

[02] Minaei-Bidgoli, B., Topchy, A., and Punch, W. (2004), A comparison of resampling methods for clustering ensembles, In Proc. International Conference on Artificial Intelligence, Vol. 2, pp. 939-945.

[03] Estabrooks, A., Jo, T., and Japkowicz, N. (2004), A Multiple Resampling Method for Learning from Imbalanced Data Sets, Comp. Intel. 20 (1) pp. 18-36.

[04] Francois, D., Rossib, F., Wertza, V., and Verleysen, M. (2007), Resampling methods for parameter-free and robust feature selection with mutual information, Neurocomputing 70(7-9):1276-1288.

[05] Good, P. (2005), Permutation, Parametric and Bootstrap Tests of Hypotheses, Springer, New York.

[06] Gretton, A., Borgwardt, K., Rasch, M., Scholkopf, B., and Smola, A. (2007), A kernel method for the two-sample- problem, In Advances in Neural Information Processing Systems (NIPS).

[07] Guo, S. (2011), Bayesian Recommender Systems: Models and Algorithms, Ph.D. thesis.

[08] Hopcroft, J., and Tarjan, R. (1973), Efficient algorithms for graph manipulation, Communications of the ACM 16: 372-378.

[09] Huang, J., Guestrin, C., and Guibas, L. (2007), Efficient Inference for Distributions on Permutations, In Advances in Neural Information Processing Systems (NIPS).

[10] Kerber, A. (1999), Applied Finite Group Actions, Springer-Verlag, Berlin.

[11] Kondor, R., Howard, A., and Jebara, T. (2007), Multi-Object Tracking with Representations of the Symmetric Group, Artificial Intelligence and Statistics (AISTATS).

[12] Kuwadekar, A. and Neville, J. (2011), Relational Active Learning for Joint Collective Classification Models, In International Conference on Machine Learning (ICML), P. 385-392.

[13] Liu, H., Palatucci, M., and Zhang, J.(2009), Blockwise coordinate descent procedures for the multi-task lasso, with applications to neural semantic basis discovery, In International Conference on Machine Learning (ICML).

[14] Matthew Higgs and John Shawe-Taylor. (2010), A PAC-Bayes bound for tailored density estimation, In Proceedings of the International Conference on Algorithmic Learning Theory (ALT).

[15] McKay, B. D. (1981), Practical graph isomorphism, Congressus Numerantium 30: 45-87, 10th. Manitoba Conf. on Numerical Math. and Computing.

[16] Mielke, P. W., and K. J. Berry (2007), Permutation Methods: A Distance Function Approach, Springer, New York.

[17] Nicholson, W. K. (2006), Introduction to Abstract Algebra, 3rd ed., Wiley, New York.

[18] Serfling, R. J. (1980), Approximation Theorems of Mathematical Statistics, Wiley, New York.

[19] Song, L. (2008), Learning via Hilbert Space Embedding of Distributions, Ph.D. thesis.

[20] Sutton, R. and Barto, A. (1998), Reinforcement Learning, MIT Press.

[21] Zhou, C., Wang, H., and Wang, Y. M. (2009), Efficient moments-based permutation tests, In Advances in Neural Information Processing Systems (NIPS), p. 2277-2285.

